# Learning the Similarity of Documents:
## An Information-Geometric Approach to Document Retrieval and Categorization

**Thomas Hofmann**
Department of Computer Science
Brown University, Providence, RI
hofmann@cs.brown.edu, www.cs.brown.edu/people/th

## Abstract

The project pursued in this paper is to develop from first information-geometric principles a general method for learning the similarity between text documents. Each individual document is modeled as a memoryless information source. Based on a latent class decomposition of the term-document matrix, a low-dimensional (curved) multinomial subfamily is learned. From this model a canonical similarity function – known as the Fisher kernel – is derived. Our approach can be applied for unsupervised and supervised learning problems alike. This in particular covers interesting cases where both, labeled and unlabeled data are available. Experiments in automated indexing and text categorization verify the advantages of the proposed method.

## 1 Introduction

The computer-based analysis and organization of large document repositories is one of today's great challenges in machine learning, a key problem being the quantitative assessment of *document similarities*. A reliable similarity measure would provide answers to questions like: How similar are two text documents and which documents match a given query best? In a time, where searching in huge on-line (hyper-)text collections like the World Wide Web becomes more and more popular, the relevance of these and related questions needs not to be further emphasized.

The focus of this work is on data-driven methods that *learn* a similarity function based on a training corpus of text documents without requiring domain-specific knowledge. Since we do not assume that labels for text categories, document classes, or topics, etc. are given at this stage, the former is by definition an *unsupervised* learning problem. In fact, the general problem of learning object similarities precedes many "classical" unsupervised learning methods like data clustering that already presuppose the availability of a metric or similarity function. In this paper, we develop a framework for learning similarities between text documents from first principles. In doing so, we try to span a bridge from the foundations of statistics in *information geometry* [13, 1] to real-world applications in *information retrieval* and text learning, namely *ad hoc* retrieval and text categorization. Although the developed general methodology is not limited to text documents, we will for sake of concreteness restrict our attention exclusively to this domain.

## 2 Latent Class Decomposition

**Memoryless Information Sources** Assume we have available a set of documents $\mathcal{D} = \{d_1, \ldots, d_N\}$ over some fixed vocabulary of words (or terms) $\mathcal{W} = \{w_1, \ldots, w_M\}$. In an information-theoretic perspective, each document $d_i$ can be viewed as an information source, *i.e.* a probability distribution over word sequences. Following common practice in information retrieval, we will focus on the more restricted case where text documents are modeled on the level of single word occurrences. This means that we we adopt the bag–of–words view and treat documents as *memoryless* information sources.[1]

> *A. Modeling assumption: Each document is a memoryless information source.*

This assumption implies that each document can be represented by a multinomial probability distribution $P(w_j|d_i)$, which denotes the (unigram) probability that a generic word occurrence in document $d_i$ will be $w_j$. Correspondingly, the data can be reduced to some simple sufficient statistics which are counts $n(d_i, w_j)$ of how often a word $w_j$ occurred in a document $d_i$. The rectangular $N \times M$ matrix with coefficients $n(d_i, w_j)$ is also called the *term-document matrix*.

**Latent Class Analysis** Latent class analysis is a decomposition technique for contingency tables (cf. [5, 3] and the references therein) that has been applied to language modeling [15] ("aggregate Markov model") and in information retrieval [7] ("probabilistic latent semantic analysis"). In latent class analysis, an unobserved class variable $z_k \in \mathcal{Z} = \{z_1, \ldots, z_K\}$ is associated with each observation, *i.e.* with each word occurrence $(d_i, w_j)$. The joint probability distribution over $\mathcal{D} \times \mathcal{W}$ is a mixture model that can be parameterized in two equivalent ways

$$P(d_i, w_j) = \sum_{k=1}^{K} P(z_k)P(d_i|z_k)P(w_j|z_k) = P(d_i)\sum_{k=1}^{K} P(w_j|z_k)P(z_k|d_i). \quad (1)$$

The latent class model (1) introduces a conditional independence assumption, namely that $d_i$ and $w_j$ are independent conditioned on the state of the associated latent variable. Since the cardinality of $z_k$ is typically smaller than the number of documents/words in the collection, $z_k$ acts as a bottleneck variable in predicting words conditioned on the context of a particular document.

To give the reader a more intuitive understanding of the latent class decomposition, we have visualized a representative subset of 16 "factors" from a $K = 64$ latent class model fitted from the Reuters21578 collection (cf. Section 4) in Figure 1. Intuitively, the learned parameters seem to be very meaningful in that they represent identifiable topics and capture the corresponding vocabulary quite well.

By using the latent class decomposition to model a collection of memoryless sources, we implicitly assume that the overall collection will help in estimating parameters for individual sources, an assumption which has been validated in our experiments.

> *B. Modeling assumption: Parameters for a collection of memoryless information sources are estimated by latent class decomposition.*

**Parameter Estimation** The latent class model has an important geometrical interpretation: the parameters $\phi_j^k \equiv P(w_j|z_k)$ define a low-dimensional subfamily of the multinomial family, $\mathcal{S}(\phi) \equiv \{\pi \in [0;1]^M : \pi_j = \sum_k \psi_k \phi_j^k$ for some $\psi \in [0;1]^K, \sum_k \psi_k = 1\}$, *i.e.* all multinomials $\pi$ that can be obtained by convex combinations from the set of "basis" vectors $\{\phi^k : 1 \le k \le K\}$. For given $\phi$–parameters,

| government | president | banks | pct | union | marks | gold | billion |
|---|---|---|---|---|---|---|---|
| tax | chairman | debt | january | air | currency | steel | dlrs |
| budget | executive | brazil | february | workers | dollar | plant | year |
| cut | chief | new | rise | strike | german | mining | surplus |
| spending | officer | loans | rose | airlines | bundesbank | copper | deficit |
| cuts | vice | dlrs | 1986 | aircraft | central | tons | foreign |
| deficit | company | bankers | december | port | mark | silver | current |
| taxes | named | bank | year | boeing | west | metal | trade |
| reform | board | payments | fell | employees | dollars | production | account |
| billion | director | billion | prices | airline | dealers | ounces | reserves |
| trading | american | trade | oil | vs | areas | food | house |
| exchange | general | japan | crude | cts | weather | drug | reagan |
| futures | motors | japanese | energy | net | area | study | president |
| stock | chrysler | ec | petroleum | loss | normal | aids | administration |
| options | gm | states | prices | mln | good | product | congress |
| index | car | united | bpd | shr | crop | treatment | white |
| contracts | ford | officials | barrels | qtr | damage | company | secretary |
| market | test | community | barrel | revs | caused | environmental | told |
| london | cars | european | exploration | profit | affected | products | volcker |
| exchanges | motor | imports | price | note | people | approval | reagans |

Figure 1: 16 selected factors from a 64 factor decomposition of the Reuters21578 collection. The displayed terms are the 10 most probable words in the class-conditional distribution $P(w_j|z_k)$ for 16 selected states $z_k$ after the exclusion of stop words.

each $\psi^i$, $\psi^i_k \equiv P(z_k|d_i)$, will define a unique multinomial distribution $\pi^i \in S(\phi)$. Since $S(\phi)$ defines a submanifold on the multinomial simplex, it corresponds to a *curved exponential subfamily.*[2] We would like to emphasis that we propose to learn both, the parameters within the family (the $\psi$'s or mixing proportions $P(z_k|d_i)$) *and* the parameters that define the subfamily (the $\phi$'s or class-conditionals $P(w_j|z_k)$).

The standard procedure for maximum likelihood estimation in latent variable models is the Expectation Maximization (EM) algorithm. In the E–step one computes posterior probabilities for the latent class variables,

$$P(z_k|d_i, w_j) \;=\; \frac{P(z_k)P(d_i|z_k)P(w_j|z_k)}{\sum_l P(z_l)P(d_i|z_l)P(w_j|z_l)} = \frac{P(z_k)P(d_i|z_k)P(w_j|z_k)}{P(d_i, w_j)} \,. \quad (2)$$

The M-step formulae can be written compactly as

$$\left. \begin{array}{c} P(d_i|z_k) \\ P(w_j|z_k) \\ P(z_k) \end{array} \right\} \propto \sum_{n=1}^{N} \sum_{m=1}^{M} n(d_n, w_m) P(z_k|d_n, w_m) \times \left\{ \begin{array}{c} \delta_{in} \\ \delta_{jm} \\ 1 \end{array} \right. , \quad (3)$$

where $\delta$ denotes the Kronecker delta.

**Related Models** As demonstrated in [7], the latent class model can be viewed as a probabilistic variant of Latent Semantic Analysis [2], a dimension reduction technique based on Singular Value Decomposition. It is also closely related to the non-negative matrix decomposition discussed in [12] which uses a Poisson sampling model and has been motivated by imposing non-negativity constraints on a decomposition by PCA. The relationship of the latent class model to clustering models like distributional clustering [14] has been investigated in [8]. [6] presents yet another approach to dimension reduction for multinomials which is based on spherical models, a different type of curved exponential subfamilies than the one presented here which is affine in the mean-value parameterization.

## 3 Fisher Kernel and Information Geometry

**The Fisher Kernel** We follow the work of [9] to derive kernel functions (and hence similarity functions) from generative data models. This approach yields a uniquely defined and intrinsic (*i.e.* coordinate invariant) kernel, called the *Fisher kernel*. One important implication is that yardsticks used for statistical models carry over to the selection of appropriate similarity functions. In spite of the purely unsupervised manner in which a Fisher kernel can be learned, the latter is also very useful in *supervised* learning, where it provides a way to take advantage of additional unlabeled data. This is important in text learning, where digital document databases and the World Wide Web offer a huge background text repository.

As a starting point, we partition the data log-likelihood into contributions from the various documents. The average log-probability of a document $d_i$, *i.e.* the probability of all the word occurrences in $d_i$ normalized by document length is given by,

$$l(d_i) = \sum_{j=1}^{M} \hat{P}(w_j|d_i) \log \sum_{k=1}^{K} P(w_j|z_k)P(z_k|d_i), \quad \hat{P}(w_j|d_i) \equiv \frac{n(d_i, w_j)}{\sum_m n(d_i, w_m)} \quad (4)$$

which is up to constants the negative Kullback-Leibler divergence between the empirical distribution $\hat{P}(w_j|d_i)$ and the model distribution represented by (1).

In order to derive the Fisher kernel, we have to compute the Fisher scores $u(d_i; \theta)$, *i.e.* the gradient of $l(d_i)$ with respect to $\theta$, as well as the Fisher information $I(\theta)$ in some parameterization $\theta$ [13]. The Fisher kernel at $\hat{\theta}$ is then given by [9]

$$\mathcal{K}(d_i, d_n) = \langle u(d_i; \hat{\theta}), I(\hat{\theta})^{-1} u(d_n; \hat{\theta}) \rangle. \quad (5)$$

**Computational Considerations** For computational reasons we propose to approximate the (inverse) information matrix by the identity matrix, thereby making additional assumptions about information orthogonality. More specifically, we use a variance stabilizing parameterization for multinomials – the square-root parameterization – which yields an isometric embedding of multinomial families on the positive part of a hypersphere [11]. In this parameterization, the above approximation will be exact for the multinomial family (disregarding the normalization constraint). We conjecture that it will also provide a reasonable approximation in the case of the subfamily defined by the latent class model.

   *C. Simplifying assumption: The Fisher information in the square-root parameterization can be approximated by the identity matrix.*

**Interpretation of Results** Instead of going through the details of the derivation which is postponed to the end of this section, it is revealing to relate the results back to our main problem of defining a similarity function between text documents. We will have a closer look at the two contributions resulting from different sets of parameters. The contribution which stems from (square-root transformed) parameters $P(z_k)$ is (in a simplified version) given by

$$\tilde{\mathcal{K}}(d_i, d_n) = \sum_k P(z_k|d_i)P(z_k|d_n)/P(z_k). \quad (6)$$

$\tilde{\mathcal{K}}$ is a weighted inner product in the low-dimensional factor representation of the documents by mixing weights $P(z_k|d_i)$. This part of the kernel thus computes a "topical" overlap between documents and is thereby able to capture *synonyms*, i.e., words with an identical or similar meaning, as well as words referring to the same

topic. Notice, that it is not required that $d_i$ and $d_n$ actually have (many) terms in common in order to get a high similarity score.

The contribution due to the parameters $P(w_j|z_k)$ is of a very different type. Again using the approximation of the Fisher matrix, we arrive at the inner product

$$\bar{\mathcal{K}}(d_i, d_n) = \sum_j \hat{P}(w_j|d_i)\hat{P}(w_j|d_n) \sum_k \frac{P(z_k|d_i, w_j)P(z_k|d_n, w_j)}{P(w_j|z_k)}. \qquad (7)$$

$\bar{\mathcal{K}}$ has also a very appealing interpretation: It essentially computes an inner product between the empirical distributions of $d_i$ and $d_n$, a scheme that is very popular in the context of information retrieval in the vector space model. However, common words only contribute, if they are explained by the same factor(s), i.e., if the respective posterior probabilities overlap. This allows to capture words with multiple meanings, so-called *polysems*. For example, in the factors displayed in Figure 1 the term "president" occurs twice (as the president of a company and as the president of the US). Depending on the document the word occurs in, the posterior probability will be high for either one of the factors, but typically not for both. Hence, the same term used in different context and different meanings will generally not increase the similarity between documents, a distinction that is absent in the naive inner product which corresponds to the degenerate case of $K = 1$.

Since the choice of $K$ determines the coarseness of the identified "topics" and different resolution levels possibly contribute useful information, we have combined models by a simple additive combination of the derived inner products. This combination scheme has experimentally proven to be very effective and robust.

*D. Modeling assumption: Similarities derived from latent class decompositions at different levels of resolution are additively combined.*

In summary, the emergence of important language phenomena like synonymy and polysemy from information-geometric principles is very satisfying and proves in our opinion that interesting similarity functions can be rigorously derived, without specific domain knowledge and based on few, explicitly stated assumptions (A-D).

**Technical Derivation**   Define $\rho_{jk} \equiv 2\sqrt{P(w_j|z_k)}$, then

$$\frac{\partial l(d_i)}{\partial \rho_{jk}} = \frac{\partial l(d_i)}{\partial P(w_j|z_k)}\frac{\partial P(w_j|z_k)}{\partial \rho_{jk}} = \sqrt{P(w_j|z_k)}\frac{\hat{P}(w_j|d_i)}{P(w_j|d_i)}P(z_k|d_i)$$

$$= \frac{\hat{P}(w_j|d_i)P(z_k|d_i, w_j)}{\sqrt{P(w_j|z_k)}}.$$

Similarly we define $\rho_k = 2\sqrt{P(z_k)}$. Applying Bayes' rule to substitute $P(z_k|d_i)$ in $l(d_i)$ (i.e. $P(z_k|d_i) = P(z_k)P(d_i|z_k)/P(d_i)$) yields

$$\frac{\partial l(d_i)}{\partial \rho_k} = \frac{\partial l(d_i)}{\partial P(z_k)}\frac{\partial P(z_k)}{\partial \rho_k} = \sqrt{P(z_k)}\frac{P(d_i|z_k)}{P(d_i)}\sum_j \frac{\hat{P}(w_j|d_i)}{P(w_j|d_i)}P(w_j|z_k)$$

$$= \frac{P(z_k|d_i)}{\sqrt{P(z_k)}}\sum_j \frac{\hat{P}(w_j|d_i)}{P(w_j|d_i)}P(w_j|z_k) \approx \frac{P(z_k|d_i)}{\sqrt{P(z_k)}}.$$

The last (optional) approximation step makes sense whenever $\hat{P}(w_j|d_i) \approx P(w_j|d_i)$. Notice that we have ignored the normalization constraints which would yield a (reactive) term that is constant for each multinomial. Experimentally, we have observed no deterioration in performance by making these additional simplifications.

|        | Medline | Cranfield | CACM | CISI |
|--------|---------|-----------|------|------|
| VSM    | 44.3    | 29.9      | 17.9 | 12.7 |
| VSM++  | 67.2    | 37.9      | 27.5 | 20.3 |

Table 1: Average precision results for the vector space baseline method (VSM) and the Fisher kernel approach (VSM++) for 4 standard test collections, Medline, Cranfield, CACM, and CISI.

|          |       | earn | acq  | money | grain | crude | average | improv. |
|----------|-------|------|------|-------|-------|-------|---------|---------|
| 20x sub  | SVM   | 5.51 | 7.67 | 3.25  | 2.06  | 2.50  | 4.20    | -       |
|          | SVM++ | 4.56 | 5.37 | 2.08  | 1.71  | 1.53  | 3.05    | +27.4%  |
|          | kNN   | 5.91 | 9.64 | 3.24  | 2.54  | 2.42  | 4.75    | -       |
|          | kNN++ | 5.05 | 7.80 | 3.11  | 2.35  | 1.95  | 4.05    | +14.7%  |
| 10x sub  | SVM   | 4.88 | 5.54 | 2.38  | 1.71  | 1.88  | 3.27    | -       |
|          | SVM++ | 4.11 | 4.84 | 2.08  | 1.42  | 1.45  | 2.78    | +15.0%  |
|          | kNN   | 5.51 | 9.23 | 2.64  | 2.55  | 2.42  | 4.47    | -       |
|          | kNN++ | 4.94 | 7.47 | 2.42  | 2.28  | 1.88  | 3.79    | +15.2%  |
| 5x sub   | SVM   | 4.09 | 4.40 | 2.10  | 1.32  | 1.46  | 2.67    | -       |
|          | SVM++ | 3.64 | 4.15 | 1.78  | 0.98  | 1.19  | 2.35    | +12.1%  |
|          | kNN   | 5.13 | 8.70 | 2.27  | 2.40  | 2.23  | 4.14    | -       |
|          | kNN++ | 4.74 | 6.99 | 2.22  | 2.18  | 1.74  | 3.57    | +13.7%  |
| all data | SVM   | 2.92 | 3.21 | 1.20  | 0.77  | 0.92  | 1.81    | -       |
| 10x cv   | SVM++ | 2.98 | 3.15 | 1.21  | 0.76  | 0.86  | 1.79    | +0.6%   |
|          | kNN   | 4.17 | 6.69 | 1.78  | 1.73  | 1.42  | 3.16    | -       |
|          | kNN++ | 4.07 | 5.34 | 1.73  | 1.58  | 1.18  | 2.78    | +12.0%  |

Table 2: Classification errors for $k$-nearest neighbors (kNN) SVMs (SVM) with the naive kernel and with the Fisher kernel(++) (derived from $K = 1$ and $K = 64$ models) on the 5 most frequent categories of the Reuters21578 corpus (earn, acq, monex-fx, grain, and crude) at different subsampling levels.

## 4   Experimental Results

We have applied the proposed method for *ad hoc* information retrieval, where the goal is to return a list of documents, ranked with respect to a given query. This obviously involves computing similarities between documents and queries. In a follow-up series of experiments to the ones reported in [7] – where kernels $\tilde{\mathcal{K}}(d_i, d_n) = \sum_k P(z_k|d_i)P(z_k|d_n)$ and $\bar{\mathcal{K}}(d_i, d_n) = \sum_j \hat{P}(w_j|d_i)\hat{P}(w_j|d_n)$ have been proposed in an *ad hoc* manner – we have been able to obtain a rigorous theoretical justification as well as some additional improvements. Average precision-recall values for four standard test collections reported in Table 1 show that substantial performance gains can be achieved with the help of a generative model (cf. [7] for details on the conducted experiments).

To demonstrate the utility of our method for supervised learning problems, we have applied it to text categorization, using a standard data set in the evaluation, the Reuters21578 collections of news stories. We have tried to boost the performance of two classifiers that are known to be highly competitive for text categorization: the $k$–nearest neighbor method and Support Vector Machines (SVMs) with a linear kernel [10]. Since we are particularly interested in a setting, where the generative model is trained on a larger corpus of unlabeled data, we have run experiments where the classifier was only trained on a subsample (at subsampling factors 20x,10x,5x). The results are summarized in Table 2. Free parameters of the base classifiers have been optimized in extensive simulations with held-out data. The results indicate

that substantial performance gains can be achieved over the standard $k$–nearest neighbor method at all subsampling levels. For SVMs the gain is huge on the subsampled data collections, but insignificant for SVMs trained on all data. This seems to indicate that the generative model does not provide any extra information, if the SVM classifier is trained on the same data. However, notice that many interesting applications in text categorization operate in the small sample limit with lots of unlabeled data. Examples include the definition of personalized news categories by just a few example, the classification and/or filtering of email, on-line topic spotting and tracking, and many more.

## 5  Conclusion

We have presented an approach to learn the similarity of text documents from first principles. Based on a latent class model, we have been able to derive a similarity function, that is theoretically satisfying, intuitively appealing, and shows substantial performance gains in the conducted experiments. Finally, we have made a contribution to the relationship between unsupervised and supervised learning as initiated in [9] by showing that generative models can help to exploit unlabeled data for classification problems.

## Footnotes

[1]Extensions to the more general case are possible, but beyond the scope of this paper.

[2]Notice that graphical models with latent variable are in general stratified exponential families [4], yet in our case the geometry is simpler. The geometrical view also illustrates the well-known identifiability problem in latent class analysis. The interested reader is referred to [3]. As a practical remedy, we have used a Bayesian approach with conjugate (Dirichlet) prior distributions over all multinomials which for the sake of clarity is not described in this paper since it is very technical but nevertheless rather straightforward.

## References

[1] Shun'ichi Amari. *Differential-geometrical methods in statistics.* Springer-Verlag, Berlin, New York, 1985.

[2] S. Deerwester, S. T. Dumais, G. W. Furnas, T. K. Landauer, and R. Harshman. Indexing by latent semantic analysis. *Journal of the American Society for Information Science*, 41:391–407, 1990.

[3] M. J. Evans, Z. Gilula, and I. Guttman. Latent class analysis of two-way contingency tables by Bayesian methods. *Biometrika*, 76(3):557–563, 1989.

[4] D. Geiger, D. Heckerman, H. King, and C. Meek. Stratified exponential families: Graphical models and model selection. Technical Report MSR-TR-98-31, Microsoft Research, 1998.

[5] Z. Gilula and S. J. Haberman. Canonical analysis of contingency tables by maximum likelihood. *Journal of the American Statistical Association*, 81(395):780–788, 1986.

[6] A. Gous. *Exponential and Spherical Subfamily Models.* PhD thesis, Stanford, Statistics Department, 1998.

[7] T. Hofmann. Probabilistic latent semantic indexing. In *Proceedings of the 22th International Conference on Research and Development in Information Retrieval (SIGIR)*, pages 50–57, 1999.

[8] T. Hofmann, J. Puzicha, and M. I. Jordan. Unsupervised learning from dyadic data. In *Advances in Neural Information Processing Systems 11*. MIT Press, 1999.

[9] T. Jaakkola and D. Haussler. Exploiting generative models in discriminative classifiers. In *Advances in Neural Information Processing Systems 11*. MIT Press, 1999.

[10] T. Joachims. Text categorization with support vector machines: Learning with many relevant features. In *International Conference on Machine Learning (ECML)*, 1998.

[11] R.E. Kass and P. W. Vos. *Geometrical foundations of asymptotic inference.* Wiley, New York, 1997.

[12] D. Lee and S. Seung. Learning the parts of objects by non-negative matrix factorization. *Nature*, 401:788–791, 1999.

[13] M. K. Murray and J. W. Rice. *Differential geometry and statistics.* Chapman & Hall, London, New York, 1993.

[14] F.C.N. Pereira, N.Z. Tishby, and L. Lee. Distributional clustering of English words. In *Proceedings of the ACL*, pages 183–190, 1993.

[15] L. Saul and F. Pereira. Aggregate and mixed–order Markov models for statistical language processing. In *Proceedings of the 2nd International Conference on Empirical Methods in Natural Language Processing*, 1997.
